# Improving Existing Fault Recovery Policies

**Guy Shani**
Department of Information Systems Engineering
Ben Gurion University, Beer-Sheva, Israel
shanigu@bgu.ac.il

**Christopher Meek**
Microsoft Research
One Microsoft Way, Redmond, WA
meek@microsoft.com

## Abstract

An automated recovery system is a key component in a large data center. Such a system typically employs a hand-made controller created by an expert. While such controllers capture many important aspects of the recovery process, they are often not systematically optimized to reduce costs such as server downtime. In this paper we describe a *passive policy learning* approach for improving existing recovery policies without exploration. We explain how to use data gathered from the interactions of the hand-made controller with the system, to create an improved controller. We suggest learning an indefinite horizon Partially Observable Markov Decision Process, a model for decision making under uncertainty, and solve it using a point-based algorithm. We describe the complete process, starting with data gathering, model learning, model checking procedures, and computing a policy.

## 1 Introduction

Many companies that provide large scale online services, such as banking services, e-mail services, or search engines, use large server farms, often containing tens of thousands of computers in order to support fast computation with low latency. Occasionally, these computers may experience failures, due to software, or hardware problems. Often, these errors can be fixed automatically through actions such as rebooting or re-imaging of the computer [6]. In such large systems it is prohibitively costly to have a technician decide on a repair action for each observed problem. Therefore, these systems often use some automatic *repair policy* or *controller* to choose appropriate repair actions.

These repair policies typically receive failure messages from the system. For example, Isard [6] suggests using a set of watchdogs — computers that probe other computers to test some attribute. Messages from the watchdogs are then typically aggregated into a small set of notifications, such as "Software Error" or "Hardware Error". The repair policy receives notifications and decides which actions can fix the observed problems. In many cases such policies are created by a human experts based on their experience and knowledge of the process. While human-made controllers often exhibit a reasonable performance, they are not automatically optimized to reduce costs. Thus, in many cases, it is possible to create a better controller, that would improve the performance of the system.

A natural choice for modeling such systems is to model each machine as a Partially Observable Markov Decision Process (POMDP) [8] — a well known model for decision making under uncertainty [12]. Given the POMDP parameters, we can compute a policy that optimizes repair costs, but learning the POMDP parameters may be difficult. Most researchers that use POMDPs therefore assume that the parameters are known. Alternatively, Reinforcement Learning (RL) [14] offers a wide range of techniques for learning optimized controllers through interactions with the environment, often avoiding the need for an explicit model. These techniques are typically used in an online learning setting, and require that the agent will *explore* all possible state-action pairs.

In the case of the management of large data centers, where inappropriate actions may result in considerable increased costs, it is unlikely that the learning process would be allowed to try every

combination of state and action. It is therefore unclear how standard RL techniques can be used in this setting. On the other hand, many systems log the interactions of the existing hand-made controller with the environment, accumulating significant data. Typically, the controller will not be designed to perform exploration, and we cannot expect such logs to contain sufficient data to train standard RL techniques.

In this paper we introduce a *passive policy learning* approach, that uses only available information without exploration, to improve an existing repair policy. We adopt the indefinite-horizon POMDP formalization [4], and use the existing controller's logs to learn the unkown model parameters, using an EM algorithm (an adapted Baum-Welch [1, 2, 15] algorithm). We suggest a model-checking phase, providing supporting evidence for the quality of the learned model, which may be crucial to help the system administrators decide whether the learned model is appropriate. We proceed to compute a policy for our learned model, that can then be used in the data center instead of the original hand-made controller.

We experiment with a synthetic, yet realistic, simulation of machine failures, showing how the policy of the learned POMDP performs close to optimal, and outperforms a set of simpler techniques that learn a policy directly in history space. We discuss the limitations of our method, mainly the dependency on a reasonable hand-made controller in order to learn good models.

Many other real world applications, such as assembly lines, medical diagnosis systems, and failure detection and recovery systems, are also controlled by hand-made controllers. While in this paper we focus on recovery from failures, our approach may be applicable to other similar domains.

## 2  Properties of the Error Recovery Problem

In this section we describe aspects of the error recovery problem and a POMDP model for the problem. Key aspects of the problem include the nature of repair actions and costs, machine failure, failure detection, and control policies.

Key aspects of repair actions include: (1) actions may succeed or fail stochastically. (2) These actions often provide an escalating behavior. We label actions using increasing *levels*, where problems fixed by an action at level $i$, are also fixed by any action of level $j > i$. Probabilistically, this would mean that if $j > i$ then $pr(healthy|a_j, e) \geq pr(healthy|a_i, e)$ for any error $e$. (3) Action costs are typically escalating, where lower level actions that fix minor problems are relatively cheap, while higher level actions are more expensive. In many real world systems this escalation is exponential. For example, restarting a service takes 5 seconds, rebooting a machine takes approximately 10 minutes, while re-imaging the machine takes about 2 hours.

Another stochastic feature of this problem is the inexact failure detection. It is not uncommon for a watchdog to report an error for a machine that is fully operational, or to report an "healthy" status for a machine that experiences a failure.

In this domain, machines are identical and independent. Typically computers in service farms share the same configuration and execute independent programs, attempting, for example, to answer independent queries to a search engine. It is therefore unlikely, if not impossible, for errors to propagate from one machine to another.

In view of the escalating nature of actions and costs, a natural choice for a policy is an *escalation policy*. Such policies choose a starting level based on the first observation, and execute an action at that level. In many cases, due to the non-deterministic success of repair actions, each action is tried several times. After the controller decides that the action at the current level cannot fix the problem, the controller escalates to the next action level. Such policies have several hand tuned decisions. For example, the number of retries of an action before an escalation occurs, and the entry level given an observation. We can hope that these features, at least, could be optimized by a learning algorithm.

System administrators typically collect logs of the hand-made controller execution, for maintenance purposes. These logs represent a valuable source of data about the system behavior that can be used to learn a policy. We would like to use this knowledge to construct an improved policy that will perform better than the original policy. Formally, we assume that we receive as input a log $L$ of repair sessions. Each repair session is a sequence $l = o_0, a_1, o_1, ..., o_{n_l}$, starting with an error notification, followed by a set of repair actions and observations until the problem is fixed. In

some cases, sessions end with the machine declared as "dead", but in practice a technician is called for these machines, repairing or replacing them. Therefore, we can assume that all sessions end successfully in the healthy state.

## 2.1 A POMDP for Error Recovery

Given the problem features above, a natural choice is to model each machine independently as a partially observable Markov decision process (POMDP) with common parameters. We define a cost-based POMDP through a tuple $< S, A, tr, C, \Omega, O >$ where $S$ is a set of states. In our case, we adopt a factored representation, where $s = < e_0, ..., e_n >$ where $e_i \in \{0, 1\}$ indicates whether error $i$ exists. That is, states are sets of failures, or errors of a machine, such as software error or a hardware failure. We also add a special state $s_H = < 0, ..., 0 >$ — the *healthy* state.

$A$ is a set of actions, such as rebooting a machine or re-imaging it. $tr(s, a, s')$ is a state transition function, specifying the probabilities of moving between states. We restrict our transition function such that $tr(s, a, s') > 0$ iff $\forall i$ if $s_i = 0$ then $s'_i = 0$. That is, an action may only fix an error, not generate new errors. $C(s, a)$ is a cost function, assigning a cost to each state-action pair. Often, costs can be measured as the time (minutes) for executing the action. For example, a reboot may take 15 minutes, while re-imaging takes 2 hours.

$\Omega$ is a set of possible observations. For us, observations are messages from the watchdogs, such as a notification of a hard disk failure, or a service reporting an error, and notifications about the success or failure of an action. $O(a, s', o)$ is an observation function, assigning a probability to each observation $pr(o|a, s')$.

In a POMDP the true state is not directly observable and we thus maintain a *belief state* $b \in B$ — a probability distribution over states, where $b(s)$ is the probability that the system is at state $s$. We assume that every repair session starts with an error observation, typically provided by one of the watchdogs. We therefore define $b_0^o$ — the prior distribution over states given an initial observation $o$. We will also maintain a probability distribution $pr_0(o)$ over initial observations. While this probability distribution is not used in model learning, it is useful for evaluating the quality of policies through trials.

It is convenient to define a policy for a POMDP as a mapping from belief states to actions $\pi : B \rightarrow A$. Our goal is to find an optimal policy that brings the machine to the healthy state with the minimal cost. One method for computing a policy is through a value function, $V$, assigning a value to each belief state $b$. Such a value function can be expressed as a set of $|S|$ dimensional vectors known as $\alpha$-vectors, i.e., $V = \{\alpha_1, ..., \alpha_n\}$. Then, $\alpha_b = \min_{\alpha \in V} \alpha \cdot b$ is the optimal $\alpha$-vector for belief state $b$, and $V(b) = b \cdot \alpha_b$ is the value that the value function $V$ assigns to $b$, where $\alpha \cdot b = \sum_i \alpha_i b_i$ is the standard vector inner product. By associating an action $a(\alpha)$ which each vector, a policy $\pi : B \rightarrow A$ can be defined through $\pi(b) = a(\alpha_b)$.

While exact value iteration, through complete updates of the belief space, does not scale beyond small toy examples, Pineau et al. [10] suggest to update the value function by creating a single $\alpha$-vector that is optimized for a specific belief state. Such methods, known as point-based value iteration, compute a value function over a finite set of belief states, resulting in a finite size value function. Perseus [13] is an especially fast point-based solver that incrementally updates a value function over a randomly chosen set of belief points, ensuring that at each iteration, the value for each belief state is improved, while maintaining a compact value function representation.

We adopt here the indefinite horizon POMDP framework [4], which we consider to be most appropriate for failure recovery. In this framework the POMDP has a single special action $a_T$, available in any state, that terminates the repair session. In our case, the action is to call a technician, deterministically repairing the machine, but with a huge cost. For example, Isard [6] estimates that a technician will fix a computer within 2 weeks. Executing $a_T$ in $s_H$ incurs no cost. Using indefinite horizon it is easy to define a lower bound on the value function using $a_T$, and execute any point-based algorithm, such as the Perseus algorithm that we use.

## 3 Learning Policies from System Logs

In this section we propose two alternatives for computing a recovery policy given the logs. We begin with a simple, model-free, history-based policy computation. Then, we suggest a more sophisticated method that learns the POMDP model parameters, and then uses the POMDP to compute a policy.

### 3.1 Model-Free Learning of $Q$-values

The optimal policy for a POMDP can be expressed as a mapping from action-observation histories to actions. Histories are directly observable, allowing us to use the standard $Q$ function terminology, where $Q(h, a)$ is the expected cost of executing action $a$ with history $h$ and continuing the session until it terminates. This approach is known as model-free, because (e.g.) the parameters of a POMDP are never learned, and has some attractive properties, because histories are directly observable, and do not require any assumption about the unobserved state space.

As opposed to standard $Q$-learning, where the $Q$ function is learned while interacting with the environment, we use the system log $L$ to compute $Q$:

$$Cost(l_i) = \sum_{j=i+1}^{|l|} C(a_j) \tag{1}$$

$$Q(h, a) = \frac{\sum_{l \in L} \delta(h + a, l) Cost(l_{|h|})}{\sum_{l \in L} \delta(h + a, l)} \tag{2}$$

where $l_i$ is a suffix of $l$ starting at action $a_i$, $C(a)$ is the cost of action $a$, $h + a$ is the history $h$ with the action $a$ appended at its end, and $\delta(h, l) = 1$ if $h$ is a prefix of $l$ and 0 otherwise. The $Q$ function is hence the average cost until repair of executing the action $a$ in history $h$, under the policy that generated $L$. Learning a $Q$ function is much faster than learning the POMDP parameters, requiring only a single pass over the training sequences in the system log.

Given the learned $Q$ function, we can define the following policy:

$$\pi_Q(h) = \min_a Q(h, a) \tag{3}$$

One obvious problem of learning a direct mapping from history to actions is that such policies do not generalize — if a history sequence was not observed in the logs, then we cannot evaluate the expected cost until the error is repaired. An approach that generalizes better is to use a finite history window of size $k$, discarding all the observations and action occurring more than $k$ steps ago. For example, when $k = 1$ the result is a completely reactive $Q$ function, computing $Q(o, a)$ using the last observation only.

### 3.2 Model-Based Policy Learning

While we assume that the behavior of a machine can be captured perfectly using a POMDP as described above, in practice we cannot expect the parameters of the POMDP to be known a-priori. In practice, the only parameters that are known are the set of possible repair actions and the set of possible observations, but even the number of possible errors is not initially known, let alone the probability of repair or observation.

Given the log of repair sessions, we can use a learning algorithm to learn the parameters of the POMDP. In this paper we choose to use an adapted Baum-Welch algorithm [1, 2, 15], an EM algorithm originally developed for computing the parameters of Hidden Markov Models (HMMs). The Baum-Welch algorithm takes as input the number of states (the number of possible errors) and a set of training sequences. Then, using the forward-backward procedure, the parameters of the POMDP are computed, attempting to maximize the likelihood of the data (the observation sequences). After the POMDP parameters have been learned, we execute Perseus [13] to compute a policy.

While training the model parameters, it is important to test likelihood on a held out set of sequences that are not used in training, in order to ensure that the resulting model does not over-fit the data. We hence split the input sequences into a train set ($80\%$) and test set ($20\%$). We check the likelihood of the test set after each forward-backward iteration, and stop the training when the likelihood of the test set does not improve.

### 3.2.1 Model Checking

When employing automatic learning methods to create an improved policy, it is important to provide evidence for the quality of the learned models. Such evidence can be helpful for the system administrators in order to make a decision whether to replace the existing policy with a new policy.

Using an imperfect learner such as Baum-Welch does not guarantee that the resulting model indeed maximizes the likelihood of the observations given the policy, even for the same policy that was used to generate the training data. Also, the loss function used for learning the model ignores action costs, thus ignoring an important aspect of the problem. For these reasons, it is possible that the resulting model will describe the domain poorly. After the model has been learned, however, we can use the average cost to provide evidence for the validity of the model. Such a process can help us determine whether these shortcomings of the learning process have indeed resulted in an inappropriate model. This phase is usually known as model checking (see, e.g. [3]).

As opposed to the $Q$-learning approach, learning a generative model (the POMDP) allows us check how similar the learned model is to the original model. We say that two POMDPs $M_1 = < S_1, A, tr_1, C_1, \Omega, O_1 >$ and $M_2 = < S_2, A, tr_2, C_2, \Omega, O_2 >$ are indistinguishable if for each policy $\pi : H \rightarrow A$, $E[\sum_t C_t | M_1, \pi] = E[\sum_t C_t | M_2, \pi]$. That is, the models are indistinguishable if any policy has the same expected accumulated cost when executed in both models.

Many policies cannot be evaluated on the real system because we cannot tolerate damaging policies. We can, however, compare the performance of the original, hand-made policy, on the system and on the learned POMDP model. We hence focus the model checking phase on comparing the expected cost of the hand-made policy predicted be the learned model to the true expected cost on the real system. To estimate the expected cost in the real system, we use the average cost of the sessions in the logs. To estimate the expected cost of the policy on the learned POMDP we execute a set of trials, each simulating a repair session, using the learned parameters of the POMDP to govern the trial advancement (observation emissions given the history and action).

We can then use the two expected cost estimates as a measure of closeness. For example, if the predicted cost of the policy over the learned POMDP is more than $20\%$ away from the true expected cost, we may deduce that the learned mode does not properly capture the system dynamics. While checking the models under a single policy cannot ensure that the models are identical, it can detect whether the model is defective. If the learned model produces a substantially different expectation over the cost of a policy than the real system, we know that the model is corrupted prior to executing its optimal policy on the real system.

After ensuring that the original policy performs similarity on the real system and on the learned model, we can also evaluate the performance of the computed policy on the learned model. Thus, we can compare the quality of the new policy to the existing one, helping us to understand the potential cost reduction of the new policy.

## 4 Empirical Evaluation

In this section we provide an empirical evaluation to demonstrate that our methods can improve an existing policy. We created a simulator of recovery sessions. In the simulator we assume that a machine can be in one of $n$ error states, or in healthy state, we also assume $n$ possible repair actions, and $m$ possible observations. We assume that each action was designed to fix a single error state, and set the number of errors to be the number of repair actions.

We set $pr(s_H | e_i, a_j) = 0.7 + 0.3 \cdot \frac{j-i}{j}$ if $j \geq i$ and 0 otherwise, simulating the escalation power of repair actions. We set $C(s, a_i) = 4^i$ and $C(s_H, a_T) = 0$, simulating the exponential growth of costs in the real AutoPilot system [6], and the zero downtime caused by terminating the session in the healthy state. For observations, we compute the relative severity of an error $e_i$ in the observation space $\mu_i = \frac{i*m}{n}$, and then set $pr(o_j | e_i, a) = \kappa e^{-\frac{(i-\mu_i)^2}{2}} / \sqrt{2\pi}$, where $\kappa$ is a normalizing factor, and $j \in [\mu_i - 1, \mu_i + 1]$.

We execute a hand-made escalation policy with 3 retries (see Section 2) over the simulator and gather a log of repair sequences. Each repair sequence begins with selecting an error uniformly, and executing the policy until the error is fixed. Then, we use the logs in order to learn a $Q$ function

Table 1: Average cost of recovery policies in simulation, with increasing model size. Results are averaged over 10 executions, and the worst standard error across all recovery policies is reported in the last column.

| Problem parameters | | | Original | Optimal | Policies learned from logs | | | | | SE |
|---|---|---|---|---|---|---|---|---|---|---|
| $|E|$ | $|O|$ | $|L|$ | $\pi_E, S$ | $\pi_{M^*}, S$ | $\pi_M, S$ | $Q, S$ | $Q_1, S$ | $Q_3, S$ | $Q_5, S$ | SE |
| 2 | 2 | 10000 | 21.6 | 17.3 | 17.3 | 17.3 | 18.0 | 17.4 | 17.3 | $< 0.2$ |
| 4 | 2 | 10000 | 220.3 | 167.7 | 172.3 | 193.6 | 174.6 | 179.6 | 190.8 | $< 3$ |
| 4 | 4 | 10000 | 221.6 | 136.8 | 141.5 | 197.8 | 239.5 | 163.6 | 178.5 | $< 2.5$ |
| 8 | 4 | 50000 | 29070 | 15047 | 20592 | 52636 | 29611 | 24611 | 27951 | $< 250$ |
| 8 | 8 | 50000 | 28978 | 15693 | 18303 | 54585 | 61071 | 26808 | 27038 | $< 275$ |

over the complete history, finite history window $Q$ functions with $k = 1, 3$, and a POMDP model. For the POMDP model, we initialize the number of states to the number of repair actions, initialize transition uniformly, and observation randomly, and execute the Baum-Welch algorithm. We also constructed a maximum-likelihood POMDP model, by initializing the state space, transition, and observation function using the true state labels (the simulated errors), and executing Baum-Welch afterwards. This initialization simulates the result of a "perfect learner" that does not suffer from the local maximum problems of Baum-Welch.

In the tables below we use $S$ for simulator, $M$ for the learned model, and $M^*$ for the model initialized by the true error labels. For policies, we use $\pi_E$ for the escalation policy, $\pi_M$ for the policy computed by Perseus on the learned model, and $\pi_{M^*}$ for the Perseus policy over the 'perfect learner' model. For the history-based $Q$ functions, we use $Q$ to denote the function computed for the complete history, and $Q_i$ denotes a policy over history suffixes of length $i$. A column header $\pi, S$ denotes the estimated cost of executing $\pi$ on the simulator, and $\pi, M$ denotes the estimated cost of executing $\pi$ on the model $M$. We also report the standard error in the estimations.

## 4.1 Results

We begin with showing the improvement of the policy of the learned models over the original escalation policies. As Table 1 demonstrates, learning a POMDP model and computing its policy always result in a substantial reduction in costs. The $M^*$ model, initialized using the true error labels, provides an upper bound on the best performance gain that can be achieved using our approach. We can see that in many cases, the result of the learned model is very closed to this upper bound.

The $Q$ functions over histories did well on the smallest domains, but not on larger domains. The worst performance is of the reactive $Q$ function ($Q_1$) over the latest observation. In the smaller domains $Q$ learning, especially with a history window of 3 ($Q_3$) does fairly well, but in the larger domains all history-based policies do not perform well.

We now take a look at the results of the model checking technique. As we explained above, a model checking phase, comparing the expected cost of a policy on both the real system and the learned model, can provide evidence as to the validity of the learned model. Indeed, as we see in Table 2, the learned models predict an expected cost that is within $3\%$ of the real expected cost.

To further validate our learned models, we also compare the expected cost of the policies computed from the models ($M$ and $M^*$) over the model and the simulator. Again, we can see that the predicted costs are very close to the real costs of these policies. As expected, the $M^*$ predicted costs are within measurement error of the true costs of the policy on the real system.

## 4.2 Discussion

The experimental results provide a few interesting insights. First, when the observation space is rich enough to capture all the errors, the learned model is very close to the optimal one. When we use fewer observations, the quality of the learned model is reduced (further from $M^*$), but the policy that is learned still significantly outperforms the original escalation policy.

It is important to note that the hand-made escalation policy that we use is very natural for domains with actions that have escalating costs and effects. Also, the number of actions and errors that we use is similar to these used by current controllers of repair services [6]. As such, the improvements

Table 2: Comparing expected cost of policies on the learned model and the simulator for model checking. Results are averaged over 10 executions, and the worst standard error across all recovery policies is reported in the last column.

| Problem parameters | | | Escalation policy | | | Optimal model | | Learned model | | |
|---|---|---|---|---|---|---|---|---|---|---|
| $|E|$ | $|O|$ | $|L|$ | $\pi_E, M^*$ | $\pi_E, M$ | $\pi_E, S$ | $\pi_{M^*}, M^*$ | $\pi_{M^*}, S$ | $\pi_M, M$ | $\pi_M, S$ | SE |
| 2 | 2 | 10000 | 21.6 | 22.3 | 21.6 | 17.3 | 17.3 | 17.6 | 17.3 | $< 0.2$ |
| 4 | 2 | 10000 | 219.5 | 227.2 | 220.4 | 165.5 | 167.7 | 173.5 | 172.4 | $< 3$ |
| 4 | 4 | 10000 | 221.1 | 225.4 | 221.6 | 137.4 | 136.8 | 138.7 | 141.5 | $< 2.5$ |
| 8 | 4 | 50000 | 28985 | 29152 | 29070 | 16461 | 15047 | 21104 | 20592 | $< 250$ |
| 8 | 8 | 50000 | 28870 | 29104 | 28978 | 15630 | 15693 | 17052 | 18303 | $< 275$ |

that we achieve over this hand-made policy hint that similar gains can be made over the real system. Our results show an improvement of $20\% - 40\%$, increasing with the size of the domain. As driving down the costs of data centers is crucial for the success of such systems, increasing performance by $20\%$ is a substantial contribution.

The history-based approaches did well only on the smallest domains. This is because for a history based value function, we have to evaluate any action at every history. As the input policy does not explore, the set of resulting histories does not provide enough coverage of the history space. For example, if the current repair policy only escalates, the history-based approach will never observe a higher level action followed by a low level action, and cannot evaluate its expected cost.

Finite history windows increase coverage, by reducing the number of possible histories to a finite scale. Thus, finite-history window provide some generalization power. Indeed, the finite history window methods (except for the reactive policy) improve upon the original escalation policy in some cases. We note, though, that we used a very simple finite history model, and more complicated approaches, such as variable length history windows [9, 11] may provide better results.

Our model checking technique indicates that none of the models that were learned, even when the number of observations was smaller than the number of errors, was defective. This is not particulary surprising, as the input data originated from a simulator that operated under the assumptions of the model. This is unlikely to happen in the real system, where any modeling assumptions compromise some aspect of the real world. Still, in the real world this technique will allow us to test, before changing the existing policy, whether the assumptions are close to the truth, and whether the model is reasonable. This is a crucial step in making the decision to replace an existing, imperfect yet operative policy, with a new one. It is unclear how to run a similar model checking phase over the history-based approach.

In improving unexploring policies, we must assume that many possible histories will not be observed. However, a complete model definition must set a value for every possibility. In learning such cases, it is important to set default values for these unknown model parameters. In our case, it is best to be pessimistic about these parameters, that is, to overestimate the cost of repair. It is therefore safe to assume that action $a$ will not fix error $e$ if we never observed $a$ to fix $e$ in the logs, except for the terminating action $a_T$.

# 5  Related Work

Using decision-theoretic techniques for troubleshooting and recovery dates back to Heckerman et al. [5], who employed Bayesian networks for troubleshooting, and a myopic approximation for recovery. Heckerman et al. assume that the parameters of the Bayesian network are given as input, and training it using the unlabeled data that the logs contain is difficult. This Bayesian network approach is also not designed for sequential data.

Partially Observable Markov Decision Processes were previously suggested for modeling automated recovery from errors. Most notably, Littman et al. [8] suggests the CSFR model which is similar to our POMDP formalization, except for a deterministic observation function, the escalation of actions, and the terminating action. They then proceed to define a belief state in this model, which is a set of possible error states, and a $Q$-function $Q(b, a)$ over beliefs. The $Q$-function is computed using standard value iteration. As these assumptions reduce the partial observability, the resulting

$Q$ function can produce good policies. Littman et al. assume that the model is either given, or that $Q$-learning can be executed online, using an exploration strategy, both of which are not applicable in our case. Also, as we argue above, in our case a $Q$ function produces substantially inferior policies because of its lack of generalization power in partially observable domains.

Another, more recent, example of a recovery approach based on POMDPs was suggested by Joshi et al. [7]. Similar to Littman et al., Joshi et al. focus on the problem of fault recovery in networks, which adds a layer of difficulty because we can no longer assume that machines are independent, as often faults cascade through the network. Joshi et al. also assume that the parameters of the model, such as the probability that a watchdog will detect each failure, and the effects of actions on failures, are known a-priori. They then suggest a one step lookahead repair strategy, and a multi-step lookahead, that uses a value function over a belief space similar to the Littman et al. belief space.

Bayer-Zubek and Dietterich [16] use a set of examples, similar to our logs, to learn a policy for disease diagnosis. They formalize the problem as an MDP, assuming that test results are discrete and exact, and use $AO*$ search, while computing the needed probabilities using the example set. They did not address the problem of missing data in the example set that arises from a non-exploring policy. Indeed, in the medical diagnosis case, one may argue that trying an action sequence that was never tried by a human doctor may result in an unreasonable risk of harm to the patient, and that therefore the system should not consider such policies.

## 6    Conclusion

We have presented an approach to improving imperfect repair policies through learning a POMDP model of the problem. Our method takes as input a log of interaction of the existing controller with the system, learns a POMDP model, and computes a policy for the POMDP that can be used in the the real system. The advantage of our method is that it does not require the existing controller to actively explore the effects of actions in all conditions, which may result in unacceptable costs in the real system. On the other hand, our approach may not converge to an optimal policy. We experiment with a synthetic, yet realistic, example of a hand-made escalation policy, where actions are ordered by increasing cost, and any action is repeated a number of times. We show how the policy of the learned model significantly improves the original escalation policy.

In the future we intend to use the improved policies to manage repairs in a real data center within the AutoPilot system [6]. The first step would be to "flight" candidate policies to evaluate their performance in the real system. Our current method is a single shot improvement, and an interesting next step is to create an incremental improvement process, where new policies constantly improve the existing one. In this setting, it would be interesting to explore bounded exploration, an exploration technique that puts a bound on the risk of the strategy.

There are a number of interesting theoretical questions about our passive policy learning method and about passive policy learning in general. First, for what families of initial policies and system dynamics would a passive policy learning method be expected to yields an improvement in expected costs. Second, what families of initial policies and systems dynamics would a passive policy learning method be expected to yield the optimal policy. Third, how would one characterize when iteratively applying a passive policy learning method would yield expected improvements in expected costs.

Finally, while this paper focuses on the important failure recovery problem, our methods may be applicable to a wide range of similar systems, such as assembly line management, and medical diagnosis systems, that currently employ hand-made imperfect controllers.

## References

[1] Leonard E. Baum, Ted Petrie, George Soules, and Norman Weiss. A maximization technique occurring in the statistical analysis of probabilistic functions of Markov chains. *The Annals of Mathematical Statistics*, 41(1):164–171, 1970.

[2] Lonnie Chrisman. Reinforcement learning with perceptual aliasing: The perceptual distinctions approach. In *In Proceedings of the Tenth National Conference on Artificial Intelligence*, pages 183–188. AAAI Press, 1992.

[3] Andrew Gelman, John B. Carlin, Hal S. Stern, and Donald B. Rubin. *Bayesian Data Analysis*. Chapman and Hall, 1996.

[4] Eric A. Hansen. Indefinite-horizon POMDPs with action-based termination. In *AAAI*, pages 1237–1242, 2007.

[5] David Heckerman, John S. Breese, and Koos Rommelse. Decision-theoretic troubleshooting. *Commun. ACM*, 38(3):49–57, 1995.

[6] Michael Isard. Autopilot: automatic data center management. *Operating Systems Review*, 41(2):60–67, 2007.

[7] Kaustubh R. Joshi, William H. Sanders, Matti A. Hiltunen, and Richard D. Schlichting. Automatic model-driven recovery in distributed systems. In *SRDS*, pages 25–38, 2005.

[8] Michael L. Littman and Nishkam Ravi. An instance-based state representation for network repair. In *in Proceedings of the Nineteenth National Conference on Artificial Intelligence (AAAI*, pages 287–292, 2004.

[9] Andrew Kachites Mccallum. *Reinforcement learning with selective perception and hidden state*. PhD thesis, 1996. Supervisor-Ballard, Dana.

[10] Joelle Pineau, Geoffrey Gordon, and Sebastian Thrun. Point-based value iteration: An anytime algorithm for POMDPs. In *International Joint Conference on Artificial Intelligence (IJCAI)*, pages 1025 – 1032, August 2003.

[11] Guy Shani and Ronen I. Brafese. Resolving perceptual aliasing in the presence of noisy sensors. In *NIPS*, 2004.

[12] R. D. Smallwood and E. J. Sondik. The optimal control of partially observable Markov decision processes over a finite horizon. *Operations Research*, 21:1071–1098, 1973.

[13] Matthijs T. J. Spaan and Nikos Vlassis. Perseus: Randomized point-based value iteration for POMDPs. *Journal of Artificial Intelligence Research*, 24:195–220, 2005.

[14] Richard S. Sutton and Andrew Barto. *Reinforcement Learning: An Introduction*. MIT Press, 1998.

[15] Daan Wierstra and Marco Wiering. Utile distinction hidden Markov models. In *ICML '04: Proceedings of the twenty-first international conference on Machine learning*, page 108, New York, NY, USA, 2004. ACM.

[16] Valentina Bayer Zubek and Thomas G. Dietterich. Integrating learning from examples into the search for diagnostic policies. *J. Artif. Intell. Res. (JAIR)*, 24:263–303, 2005.

